# CAM Storage of Analog Patterns and Continuous Sequences with $3N^2$ Weights

**Bill Baird**
Dept Mathematics and
Dept Molecular and Cell Biology,
129 LSA, U.C.Berkeley,
Berkeley, Ca. 94720

**Frank Eeckman**
Lawrence Livermore
National Laboratory,
P.O. Box 808 (L-426),
Livermore, Ca. 94550

## Abstract

A simple architecture and algorithm for analytically guaranteed associative memory storage of analog patterns, continuous sequences, and chaotic attractors in the same network is described. A matrix inversion determines network weights, given prototype patterns to be stored. There are $N$ units of capacity in an $N$ node network with $3N^2$ weights. It costs one unit per static attractor, two per Fourier component of each sequence, and four per chaotic attractor. There are no spurious attractors, and there is a Liapunov function in a special coordinate system which governs the approach of transient states to stored trajectories. Unsupervised or supervised incremental learning algorithms for pattern classification, such as competitive learning or bootstrap Widrow-Hoff can easily be implemented. The architecture can be "folded" into a recurrent network with higher order weights that can be used as a model of cortex that stores oscillatory and chaotic attractors by a Hebb rule. Hierarchical sensory-motor control networks may be constructed of interconnected "cortical patches" of these network modules. Network performance is being investigated by application to the problem of real time handwritten digit recognition.

## 1  Introduction

We introduce here a "projection network" which is a new network for implementation of the "normal form projection algorithm" discussed in [Bai89, Bai90b]. The autoassociative case of this network is formally equivalent to the previous higher order network realization used as a biological model [Bai90a]. It has $3N^2$ weights instead of $N^2 + N^4$, and is more useful for engineering applications. All the mathematical results proved for the projection algorithm in that case carry over to this

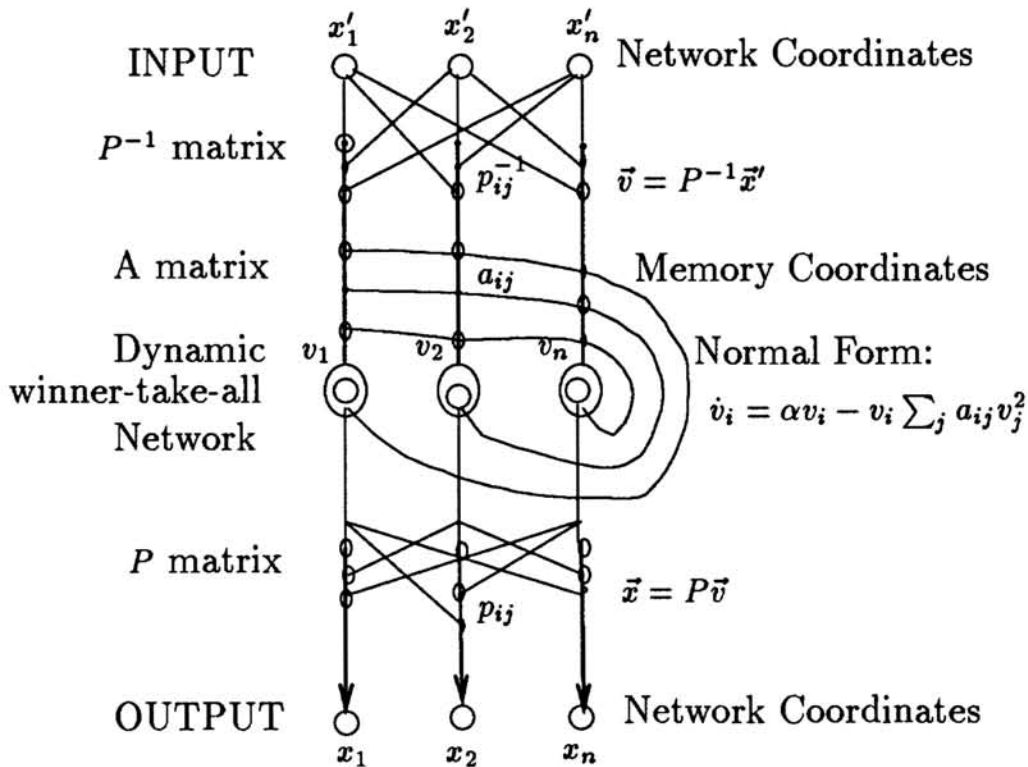

INPUT — $x'_1$ $x'_2$ $x'_n$ — Network Coordinates

$P^{-1}$ matrix — $p_{ij}^{-1}$ — $\vec{v} = P^{-1}\vec{x}'$

A matrix — $a_{ij}$ — Memory Coordinates

Dynamic winner-take-all Network — $v_1$ $v_2$ $v_n$ — Normal Form:

$$\dot{v}_i = \alpha v_i - v_i \sum_j a_{ij} v_j^2$$

$P$ matrix — $p_{ij}$ — $\vec{x} = P\vec{v}$

OUTPUT — $x_1$ $x_2$ $x_n$ — Network Coordinates

Figure 1: Projection Network - $3N^2$ weights. The $A$ matrix determines a k-winner-take-all net - programs attractors, basins of attraction, and rates of convergence. The columns of $P$ contain the ouput patterns associated to these attractors. The rows of $P^{-1}$ determine category centroids

new architecture, but more general versions can be trained and applied in novel ways. The discussion here will be informal, since space prohibits technical detail and proofs may be found in the references above.

A key feature of a net constructed by this algorithm is that the underlying dynamics is explicitly isomorphic to any of a class of standard, well understood nonlinear dynamical systems - a "normal form" [GH83]. This system is chosen in advance, independent of both the patterns to be stored and the learning algorithm to be used. This control over the dynamics permits the design of important aspects of the network dynamics independent of the particular patterns to be stored. Stability, basin geometry, and rates of convergence to attractors can be programmed in the standard dynamical system.

Here we use the normal form for the Hopf bifurcation [GH83] as a simple recurrent competitive k-winner-take-all network with a cubic nonlinearity. This network lies in what might considered diagonalized or "overlap" or "memory coordinates" (one memory per k nodes). For temporal patterns, these nodes come in complex conjugate pairs which supply Fourier components for trajectories to be learned. Chaotic dynamics may be created by specific programming of the interaction of two pairs of these nodes.

Learning of desired spatial or spatio-temporal patterns is done by projecting sets of

these nodes into network coordinates(the standard basis) using the desired vectors as corresponding columns of a transformation matrix $P$. In previous work, the differential equations of the recurrent network itself are linearly transformed or "projected", leading to new recurrent network equations with higher order weights corresponding to the cubic terms of the recurrent network.

## 2   The Projection Network

In the projection net for autoassociation, this algebraic projection operation into and out of memory coordinates is done explicitly by a set of weights in two feed-forward linear networks characterized by weight matrices $P^{-1}$ and $P$. These map inputs into and out of the nodes of the recurrent dynamical network in memory coordinates sandwiched between them. This kind of network, with explicit input and output projection maps that are inverses, may be considered an "unfolded" version of the purely recurrent networks described in the references above.

This network is shown in figure 1. Input pattern vectors $\vec{x}'$ are applied as pulses which project onto each vector of weights (row of the $P^{-1}$ matrix) on the input to each unit $i$ of the dynamic network to establish an activation level $v_i$ which determines the initial condition for the relaxation dynamics of this network. The recurrent weight matrix $A$ of the dynamic network can be chosen so that the unit or predefined subspace of units which recieves the largest projection of the input will converge to some state of activity, static or dynamic, while all other units are supressed to zero activity.

The evolution of the activity in these memory coordinates appears in the original network coordinates at the output terminals as a spatio-temporal pattern which may be fully distributed accross all nodes. Here the state vector of the dynamic network has been transformed by the $P$ matrix back into the coordinates in which the input was first applied. At the attractor $\vec{v}*$ in memory coordinates, only a linear combination of the columns of the $P$ weight matrix multiplied by the winning nonzero modes of the dynamic net constitute the network representation of the output of the system. Thus the attractor retrieved in memory coordinates reconstructs its learned distributed representation $\vec{x}^*$ through the corresponding columns of the output matrix $P$, e.g.   $P^{-1}\vec{x}' = \vec{v}$ ,   $\vec{v} \to \vec{v}^*$ ,   $P\vec{v}^* = \vec{x}^*$ .

For the special case of content addressable memory or autoassociation, which we have been describing here, the actual patterns to be learned form the columns of the output weight matrix $P$, and the input matrix is its inverse $P^{-1}$. These are the networks that can be "folded" into higher order recurrent networks. For orthonormal patterns, the inverse is the transpose of this output matrix of memories, $P^{-1} = P^T$, and no computation of $P^{-1}$ is required to store or change memories - just plug the desired patterns into appropriate rows and columns of $P$ and $P^T$.

In the autoassociative network, the input space, output space and normal form state space are each of dimension $N$. The input and output linear maps require $N^2$ weights each, while the normal form coefficients determine another $N^2$ weights. Thus the net needs only $3N^2$ weights, instead of the $N^2 + N^4$ weights required by the folded recurrent network. The $2N^2$ input and output weights could be stored off-chip in a conventional memory, and the fixed weights of the dynamic normal form network could be implemented in VLSI for fast analog relaxation.

## 3    Learning Extensions

More generally, for a heteroassociative net (i. e., a net designed to perform a map from input space to possibly different output space) the linear input and output maps need not be inverses, and may be noninvertible. They may be found by any linear map learning technique such as Widrow-Hoff or by finding pseudoinverses.

Learning of all desired memories may be instantaneous, when they are known in advance, or may evolve by many possible incremental methods, supervised or unsupervised. The standard competitive learning algorithm where the input weight vector attached to the winning memory node is moved toward the input pattern can be employed. We can also decrease the tendency to choose the most frequently selected node, by adjusting paratmeters in the normal form equations, to realize the more effective frequency selective competitive learning algorithm [AKCM90]. Supervised algorithms like bootstrap Widrow Hoff may be implemented as well, where a desired output category is known. The weight vector of the winning normal form node is updated by the competitive rule, if it is the right category for that input, but moved away from the input vector, if it is not the desired category, and the weight vector of the desired node is moved toward the input.

Thus the input map can be optimized for clustering and classification by these algorithms, as the weight vectors (row vectors of the input matrix) approach the centroids of the clusters in the input environment. The output weight matrix may then be constructed with any desired output pattern vectors in appropriate columns to place the attractors corresponding to these categories anywhere in the state space in network coordinates that is required to achieve a desired heteroassociation.

If either the input or the output matrix is learned, and the other chosen to be its inverse, then these competitive nets can be folded into oscillating biological versions, to see what the competive learning algorithms correspond to there. Now either the rows of the input matrix may be optimized for recognition, or the columns of the output matrix may be chosen to place attractors, but not both. We hope to be able to derive a kind of Hebb rule in the biological network, using the unfolded form of the network, which we can prove will accomplish competitive learning. Thus the work on engineering applications feeds back on the understanding of the biological systems.

## 4    Programming the Normal Form Network

The key to the power of the projection algorithm to program these systems lies in the freedom to chose a well understood normal form for the dynamics, independent of the patterns to be learned. The Hopf normal form used here, (in Cartesian coordinates)   $\dot{v}_i = \sum_{j=1}^{N} J_{ij} v_j - v_i \sum_{j=1}^{N} A_{ij} v_j^2$   is especially easy to work with for programming periodic attractors, but handles fixed points as well. $J$ is a matrix with real eigenvalues for determining static attractors, or complex conjugate eignevalue pairs in blocks along the diagonal for periodic attractors. The real parts are positive, and cause initial states to move away from the origin, until the competitive (negative) cubic terms dominate at some distance, and cause the flow to be inward from all points beyond. The off-diagonal cubic terms cause competition between directions of flow within a spherical middle region and thus create multiple attractors and basins. The larger the eigenvalues in $J$ and off-diagonal weights in

$A$, the faster the convergence to attractors in this region.

It is easy to choose blocks of coupling along the diagonal of the $A$ matrix to produce different kinds of attractors, static, periodic, or chaotic, in different coordinate subspaces of the network. The sizes of the subspaces can be programmed by the sizes of the blocks. The basin of attraction of an attractor determined within a subspace is guaranteed to contain the subspace [Bai90b]. Thus basins can be programmed, and "spurious" attractors can be ruled out when all subspaces have been included in a programmed block.

This can be accomplished simply by choosing the $A$ matrix entries outside the blocks on the diagonal (which determine coupling of variables within a subspace) to be greater (more negative) than those within the blocks. The principle is that this makes the subspaces defined by the blocks compete exhaustively, since intersubspace competition is greater than subspace self-damping. Within the middle region, the flow is forced to converge laterally to enter the subspaces programmed by the blocks.

An simple example is a matrix of the form,

$$
A = \begin{bmatrix}
d & & & & & & & \\
& d & & & & & (g) & \\
& & \begin{bmatrix} d & c \\ c & d \end{bmatrix} & & & & & \\
& & & \begin{bmatrix} d & d & c & c \\ d & d & c & c \\ c & c & d & d \\ c & c & d & d \end{bmatrix} & & \\
& & (g) & & & & & \ddots
\end{bmatrix},
$$

where $0 < c < d < g$. There is a static attractor on each axis (in each one dimensional subspace) corresponding to the first two entries on the diagonal, by the agrument above. In the first two dimensional subspace block there is a single fixed point in the interior of the subspace on the main diagonal, because the off-diagonal entries within the block are symmetric and less negative than those on the diagonal. The components do not compete, but rather combine. Nevertheless, the flow from outside is into the subspace, because the entries outside the subspace are more negative than those within it.

The last subspace contains entries appropriate to guarantee the stability of a periodic attractor with two frequencies (Fourier components) chosen in the $J$ matrix. The doubling of the entries is because these components come in complex conjugate pairs (in the $J$ matrix blocks) which get identical $A$ matrix coupling. Again, these pairs are combined by the lesser off-diagonal coupling within the block to form a single limit cycle attractor. A large subspace can store a complicated continuous periodic spatio-temporal sequence with many component frequencies.

The discrete Fourier transform of a set of samples of such a sequence in space and time can be input directly to the $P$ matrix as a set of complex columns corresponding to the frequencies in $J$ and the subspace programmed in $A$. $N/2$ total DFT samples of $N$ dimensional time varying spatial vectors may be placed in the $P$ matrix, and parsed by the $A$ matrix into $M < N/2$ separate sequences as desired, with separate basins of attraction guaranteed [Bai90b]. For a symmetric $A$ matrix, there is a

Liapunov function, in the amplitude equations of a polar coordinate version of the normal form, which governs the approach of initial states to stored trajectories.

## 5    Chaotic Attractors

Chaotic attractors may be created in this normal form, with sigmoid nonlinearities added to the right hand side, $v_i \rightarrow tanh(v_i)$. The sigmoids yield a spectrum of higher order terms that break the phase shift symmetry of the system. Two oscillatory pairs of nodes like those programmed in the block above can then be programmed to interact chaotically. In our simulations, for example, if we set the upper block of $d$ entries to -1, and the lower to 1, and replace the upper $c$ entries with 4.0, and the lower with -0.4, we get a chaotic attractor of dimension less than four, but greater than three.

This is "weak" or "phase coherent" chaos that is still nearly periodic. It is created by the broken symmetry, when a homoclinic tangle occurs to break up an invariant 3-torus in the flow [GH83]. This is the Ruelle-Takens route to chaos and has been observed in Taylor-Couette flow when both cylnders are rotated. We believe that sets of Lorentz equations in three dimensional subspace blocks could be used in a projection network as well. Experiments of Freeman, however, have suggested that chaotic attractors of the above dimension occur in the olfactory system [Fre87]. These might most naturally occur by the interaction of oscillatory modes.

In the projection network or its folded biological version, these chaotic attractors have a basin of attraction in the $N$ dimensional state space that constitues a category, just like any other attractor in this system. They are, however, "fuzzy" attractors, and there may be computational advantages to the basins of attraction (categories) produced by chaotic attractors, or to the effects their outputs have as fuzzy inputs to other network modules. The particular $N$ dimensional spatio-temporal patterns learned for the four components of these chaotically paired modes may be considered a coordinate specific "encoding" of the strange attractor, which may constitute a recognizable input to another network, if it falls within some learned basin of attraction. While the details of the trajectory of a strange attractor in any real physical continuous dynamical system are lost in the noise, there is still a particular statistical structure to the attractor which is a recognizable "signature".

## 6    Applications

Handwritten characters have a natural translation invariant analog representation in terms of a sequence of angles that parametrize the pencil trajectory, and their classification can be taken as a static or temporal pattern recognition problem. We have constructed a trainable on-line system to which anyone may submit input by mouse or digitizing pad, and observe the performance of the system for themselves, in immediate comparison to their own internal recognition response. The performance of networks with static, periodic, and chaotic attractors may be tested simultaneously, and we are presently assessing the results.

These networks can be combined into a hierarchical architecture of interconnected modules. The larger network itself can then be viewed as a projection network, transformed into biological versions, and its behavior analysed with the same tools that were used to design the modules. The modules can model "patches" of cortex

interconnected to form sensory-motor control networks. These can be configured to yield autonomous adaptive "organisms" which learn useful sequences of behaviors by reinforcement from their environment.

The *A* matrix for a network like that above may itself become a sub-block in the *A* matrix of a larger network. The overall network is then a projection network with zero elements in off-diagonal *A* matrix entries outside blocks that define multiple attractors for the submodules. The modules neither compete nor combine states, in the absence of *A* matrix coupling between them, but take states independently based on their inputs to each other through the weights in the matrix *J* (which here describes full coupling). The modules learn connection weights $J_{ij}$ between themselves which will cause the system to evolve under a clocked "machine cycle" by a sequence of transitions of attractors (static, oscillatory, or chaotic) within the modules, much as a digital computer evolves by transitions of its binary flip-flop states. This entire network may be folded to use more fault tolerant and biologically plausible distributed representations, without disrupting the identity of the subnetworks.

Supervised learning by recurrent back propagation or reinforcement can be used to train the connections between modules. When the inputs from one module to the next are given as impulses that establish initial conditions, the dynamical behavior of a module is described exactly by the projection theorem [Bai89]. Possible applications include problems such as system identification and control, robotic path planning, gramatical inference, and variable-binding by phaselocking in oscillatory semantic networks.

**Acknowledgements:**

Supported by AFOSR-87-0317, and a grant from LLNL. It is a pleasure to acknowledge the support of Walter Freeman and invaluable assistance of Morris Hirsch.

# References

[AKCM90] C. Ahalt, A. Krishnamurthy, P. Chen, and D. Melton. Competitive learning algorithms for vector quantization. *Neural Networks*, 3:277–290, 1990.

[Bai89] B Baird. A bifurcation theory approach to vector field programming for periodic attractors. In *Proc. Int. Joint Conf. on Neural Networks, Wash. D.C.*, pages 1:381–388, June 1989.

[Bai90a] B. Baird. Bifurcation and learning in network models of oscillating cortex. In S. Forest, editor, *Emergent Computation*, pages 365–384. North Holland, 1990. also in Physica D, 42.

[Bai90b] B. Baird. A learning rule for cam storage of continuous periodic sequences. In *Proc. Int. Joint Conf. on Neural Networks, San Diego*, pages 3: 493–498, June 1990.

[Fre87] W.J. Freeman. Simulation of chaotic eeg patterns with a dynamic model of the olfactory system. *Biological Cybernetics*, 56:139, 1987.

[GH83] J. Guckenheimer and D. Holmes. *Nonlinear Oscillations, Dynamical Systems, and Bifurcations of Vector Fields*. Springer, New York, 1983.